# Nonlinear Filtering of Electron Micrographs by Means of Support Vector Regression

**R. Vollgraf[1], M. Scholz[1], I. A. Meinertzhagen[2], K. Obermayer[1]**
[1]Department of Electrical Engineering and Computer Science
Berlin University of Technology, Germany
{vro,idefix,oby}@cs.tu-berlin.de
[2]Dalhousie University, Halifax, Canada
iam@is.dal.ca

## Abstract

Nonlinear filtering can solve very complex problems, but typically involve very time consuming calculations. Here we show that for filters that are constructed as a RBF network with Gaussian basis functions, a decomposition into linear filters exists, which can be computed efficiently in the frequency domain, yielding dramatic improvement in speed. We present an application of this idea to image processing. In electron micrograph images of photoreceptor terminals of the fruit fly, *Drosophila*, synaptic vesicles containing neurotransmitter should be detected and labeled automatically. We use hand labels, provided by human experts, to learn a RBF filter using Support Vector Regression with Gaussian kernels. We will show that the resulting nonlinear filter solves the task to a degree of accuracy, which is close to what can be achieved by human experts. This allows the very time consuming task of data evaluation to be done efficiently.

## 1 Introduction

Using filters for image processing can be understood as a supervised learning method for classification and segmentation of certain image elements. A given training image would contain a target that should be approximated by some filter at every location. In principle, any kind of machine-learning techniques could be employed to learn the mapping from the input receptive field of the filter to the target value. The most simple filter is linear mapping. It has the advantage that it can be very efficiently computed in the frequency domain. However linear filters may not be complex enough for difficult problems. The complexity of nonlinear filters is in principle unlimited (if we leave generalization issues aside), but the computation of the filter output can be very time consuming, since usually there is no shortcut in the frequency domain, as for linear filters. However, for nonlinear filters, that are linear superpositions of Gaussian radial basis functions, there exists a decomposition into linear filters, allowing the filter output to be computed in reasonable

time. This sort of nonlinear filtering is for example obtained, when Support Vector Machines (SVM) with a Gaussian kernel are used for learning. SVM have proved to yield good performance on many applications [1]. This and the ability to compute the filter output in an affordable time, make SVM interesting for nonlinear filtering in image processing tasks. Here we apply this new method to the evaluation of electron micrograph images taken from the visual system of the fruit fly, *Drosophila*, as a means to analyze morphological phenotypes of new genetic mutants. Genetically manipulable organisms such as *Drosophila* provide means to address many current questions in neuroscience. The action, even of lethal genes, can be uncovered in photoreceptors by creating homozygous whole-eye mosaics in heterozygous flies [2]. Mutant synaptic phenotypes are then interpretable from detailed ultra-structural knowledge of the photoreceptor terminals R1-R6 in the lamina [3]. Electron microscopy (EM) alone offers the resolution required to analyze sub-cellular structure, even though this technique is tedious to undertake. In *Drosophila* genetics hundreds of mutants of the visual system have been isolated, many even from a single genetic screen. The task of analyzing each of these mutants manually is simply not feasible, hence reliable automatic (computer assisted) methods are needed. The focus here is just to count the number of synaptic vesicles, but in general the method proposed in this report could be extended to the analysis of other structures as well.

As representative datasets showing the feasibility of the proposed method, we have chosen two datasets from wild type *Drosophila* (`ter01` for training and `ter04` for performance evaluation, cf. Fig. 1) and one from a visual system mutant (`mutant`, also for performance evaluation, cf. Fig. 2, left).

## 2 Learning the RBF Filter

Given an image $x$, we want to find a RBF filter with Gaussian basis functions, the output of which is closest to a target image $y$, in terms of some suitable distance measure. The filter is constrained to some receptive field $P$, so that its output at position $r$ would be formulated in the most general form as

$$z(r) = f_{RBF}(\mathbf{x}(r)) = f_{RBF}\left((x(r + \Delta r_1), \ldots, x(r + \Delta r_M))^T\right) ,$$

where $P = \{\Delta r_1, \ldots, \Delta r_M\}$ is the neighborhood that forms the receptive field. In the following we will continue using bold faced symbols to indicate a vector containing the neighborhood (patch) at some location, while light faces indicate the value of the image itself. Individual elements of patches are addressed by a subscript, for example $\mathbf{x}_{\Delta r}(r) = x(r + \Delta r)$. $f_{RBF}$ is a RBF network with $M$ input dimensions. It can be implemented as a feed forward net with a single hidden layer containing a fixed number of RBF units and a linear output layer [4]. However we would rather use the technique of *Support Vector Regression (SVR)* [5] as it has a number of advantages over RBF feed forward networks. It offers adjustable model complexity depending on the training data, thus providing good generalization performance. The training of SVR is a quadratic, constrained optimization problem, which can be solved efficiently without being trapped into local minima. In the linear case the formulation of the $\nu-$SVR, as it was introduced in [6], would be

$$\text{minimize} \quad \tau(\mathbf{w}, \xi^{(*)}, \varepsilon) = \frac{1}{2}\|\mathbf{w}\|^2 + C \cdot \left(\nu\varepsilon + \frac{1}{l}\sum_{i=1}^{l}(\xi_i + \xi_i^*)\right) \quad (1)$$

$$\text{s.t.} \quad ((\mathbf{w} \cdot \mathbf{x}_i) + b) - y_i \leq \varepsilon + \xi_i , \qquad y_i - ((\mathbf{w} \cdot \mathbf{x}_i) + b) \leq \varepsilon + \xi_i^* \quad (2)$$

$$\xi_i^{(*)} \geq 0, \varepsilon \geq 0 \quad (3)$$

The constraints implement as a distance measure the $\varepsilon$-insensitive loss $|y - f(\mathbf{x})|_\varepsilon = \max\{0, |y - f(\mathbf{x})| - \varepsilon\}$, which is a basic feature of SVR, and has been shown to yield robust estimation. The objective itself provides a solution of low complexity (small $\|\mathbf{w}\|^2$) and, at the same time, low errors, balanced by $C$. In contrast to $\varepsilon$−SVR, as it was introduced at first in [5], parameterization with the hyper parameter $\nu$ also allows optimization for the width $\varepsilon$ of the insensitive region. Interacting with $C$, $\nu$ controls the complexity of the model. It provides an upper bound on the fraction of outliers (samples that do not fall into the epsilon tube) and a lower bound on the fraction of support vectors (SV), see [6] and [1] for further details. As usual for SVM, the system is transformed into a nonlinear regressor by replacing the scalar product with a kernel, that fulfills Mercers condition [7]. With a Gaussian kernel (RBF kernel) the regression function is

$$z(r) = \sum_{i=1}^{l} \alpha_i^{(*)} z_i(r) + b \ , \tag{4}$$

where

$$z_i(r) = k(\mathbf{x}_i, \mathbf{x}(r)) = \exp\left(-\frac{1}{\gamma} \sum_{\Delta r \in P} (\mathbf{x}_{i,\Delta r} - x(r + \Delta r))^2\right) \tag{5}$$

is the Gaussian- or RBF-kernel. The resulting SVs $\mathbf{x}_i$ are a subset of the training examples, for which one of the constraints (2) holds with equality. They correspond to Lagrange multipliers having $\alpha_i^{(*)} = (\alpha_i - \alpha_i^*) \neq 0$. In the analogy to a RBF network, the SVs are the centers of the basis functions, while $\alpha_i^{(*)}$ are the weights of the output layer.

## 3 RBF Filtering

To evaluate a RBF network filter at location $r$, all the basis functions have to be evaluated for the neighborhood $\mathbf{x}(r)$. This calculation is computationally very expensive when computed in the straightforward way given by (5). If the squared sum is multiplied out, however, we can compute the kernel as

$$z_i(r) = \exp\left(-\frac{1}{\gamma} \left(\|\mathbf{x}_i\|^2 - 2z_i'(r) + z_i''(r)\right)\right) \ , \tag{6}$$

where

$$z_i'(r) = \sum_{\Delta r \in P} \mathbf{x}_{i,\Delta r} x(r + \Delta r) \quad \text{and} \quad z_i''(r) = \sum_{\Delta r \in P} x(r + \Delta r)^2 \ . \tag{7}$$

Now we are left with linear filtering operations only, the two cross correlations $z'$ and $z''$, which can be efficiently computed in the frequency domain, where the cross correlation of a signal with some filter becomes a multiplication of the signal's spectrum with the conjugate complex spectrum of the filter. This operation is so much faster that the additional computation cost of the Fourier transform is neglectable. Note that in fact $z''$ is the cross correlation of $x^2$ with the filter $\mathbf{o}$, which is 1 for all $\Delta r \in P$. We need to compute the following Fourier transforms:

$$\begin{aligned}
X(j\omega) &\equiv \mathcal{F}[x(r)] \ , & X^{(2)}(j\omega) &\equiv \mathcal{F}[x^2(r)] \ , \\
X_i(j\omega) &\equiv \mathcal{F}[x_i(r)] \ , & O(j\omega) &\equiv \mathcal{F}[o(r)] \ .
\end{aligned} \tag{8}$$

$x_i(r)$ and $o(r)$ are the filters $\mathbf{x}_i$ and $\mathbf{o}$, zero filled for $r \notin P$ to the size of the image. It is necessary to take care of the placement of the origin $\Delta r = 0$ and the mapping of negative offsets in $P$, which depends on the implementation of the Fourier transform. Now $z_i$ is easily computed as

$$z_i(r) \equiv \exp\left(-\frac{1}{\gamma}\left(\mathbf{x}_i^T\mathbf{x}_i - \mathcal{F}^{-1}\left[2X_i^C(j\omega)X(j\omega) - O^C(j\omega)X^{(2)}(j\omega)\right]\right)\right) \quad (9)$$

where $(\cdot)^C$ indicates the conjugate complex. Using *Fast Fourier Transform (FFT)*, the speed improvement is much higher when the size of $x$ is even in terms of powers of 2 [8]. Thus one should consider enlarging the image size by adding the appropriate number of zeros at the border. However this can lead to large overhead regions, when the image size is not close to the next power of 2. For this reason we use a tiling scheme, which processes the image in smaller parts of even size, which can cover the entire image more closely. It is important to be aware of the distorted margins of the image or its tiles, when filtering is done in the frequency domain. Because the cross correlation in the frequency domain is cyclic, points at the margin, for which the neighborhood $P$ exceeds the image boundaries, have incorrect values in the filter's output. This is particularly important for the tiling scheme, which has to provide sufficient overlap for the tiles, so that the image can be covered completely with the uncorrupted inner parts of the tiles. Table 1 summarizes the speed-up gain for the described filtering method. Most performance gain is obtained through the filtering in the frequency domain. However, splitting the image into tiles of appropriate size can improve speed even further.

Table 1: Computation time examples for different filtering methods.

| filtering acc. to (5) | 6d 10h |
|---|---|
| FFT filtering, whole image | 55m |
| FFT filtering, tiles of $256 \times 256$ | 24m |

- image size $1686 \times 1681$ pixel
- 200 SV of $50 \times 50$ pixels size
- implementation in *MATLAB*
- *SUN F6800 / 750MHz, 1 CPU*

## 4    Experiments

To test the performance of the method we used two images of wild type and one of mutant photoreceptor terminals. The profiles of the terminals contain typically about 100 synaptic vesicles, the number of which could differ if the genes for membrane trafficking are mutated. Detecting such numerical differences is a simple but tedious task best suited to a computational approach. The wild type images came from electron micrographs of the same animal under the same physiological conditions. For all images visual identification and hand written labelings of the vesicles were made. Image ter01 (Fig. 1, left) was used for training. The validation error on ter04 (Fig. 1, right) was considered for model selection. Then the best model was tested on the mutant image (Fig. 2).

### 4.1    Construction of the Target

ter01 contains 286 hand-labeled vesicles at discrete positions. To generate a smooth target image $y$, circular gauss blobs with $\sigma^2 = 40$ and a peak value of 1 were placed on every label. Now training examples $\mathbf{x}(r)$ where generated from ter01 by

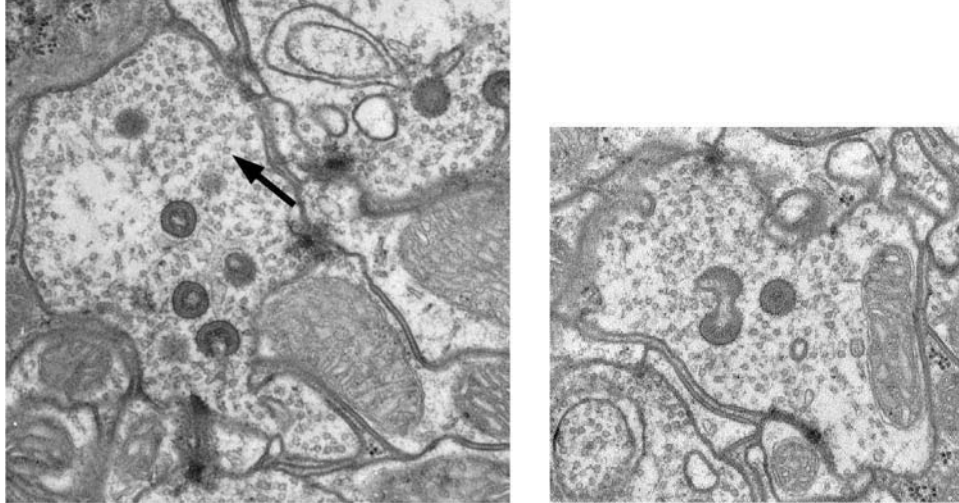

Figure 1: EM images of photoreceptor terminals of the wild type fruit fly, *Drosophila melanogaster*. The left image (`ter01`) was used for training, the right image (`ter04`) for validation. *Arrow:* individual synaptic vesicle, 30nm in diameter.

taking square patches, centered around $r$. We have set the patch size $P = 50 \times 50$ pixels, to cover an entire vesicle plus a little surrounding. The corresponding values $y(r)$ of the target image where used as targets for regression. The most complete training set would clearly contain patches from all locations, which however would be computationally unfeasible. Instead we used patches from all hand-label positions and additionally 2000 patches from random positions. No patches exceeded the image boundaries. With these data the SVM was trained. We used the `libsvm` implementation [9] which also contains, beside others, the $\nu$-SVR. Mainly three parameters have to be adjusted for training the $\nu$-SVR: the width of the RBF kernel $\gamma$ and the parameters $\nu$ and $C$. Since the training dataset is small compared to the input dimensionality, the validation error on `ter04` is subject to large variance. Therefore we cannot give a complete parameter exploration here, but we would expect a model with not too much complexity to give the best generalization. It turned out that, for the given conditions, a kernel size of $\gamma = 20.000$ together with a low value $\nu = 0.1$ and $C = 0.01$ yield good validation results on `ter04`. The optimization returned 245 SVs, 185 of which where outliers. The kernel width is large compared with the average distance of the training examples in input space, which was $< 2.000$. Because the computation time of the filter grows linearly with the number of SVs, we are strongly interested in a solution with only few SVs. This requires small values of $\nu$, since it is a lower bound on the fraction of SVs. At the same time, small $\nu$ values provide large $\varepsilon$ and hence restrict the model complexity. After filtering, the decision which point in $z$ corresponds to a vesicle, has to be made. Although the regions of high amplitude form sharp peaks, they still have some spatial extension. Therefore we first discriminate for the peak locations and then for the amplitude. In a first step, we determine those locations $r$, for which $z(r)$ is a local maximum in some neighborhood, which is determined roughly by the size of a vesicle, i.e. we consider the set

$$Q_d = \left\{ r : \ z(r) = \max_{\{\Delta r : \|r - \Delta r\| \leq d\}} z(r + \Delta r) \right\} \ . \tag{10}$$

Then a threshold is applied to the candidates in $Q_d$ to yield the set of locations, which are considered as detected vesicles,

$$Q_\theta = \{r \in Q_d : z(r) > \theta\} \ . \tag{11}$$

We set the parameter $d = 15$ constant in our experiments, and will vary only the threshold $\theta$.

## 4.2 Performance Evaluation

To evaluate the performance of the method, the set of detected vesicles $Q_\theta$ must be compared with set $Q_{Exp}$, which contains the locations detected by a human expert. Clearly this is only meaningful when done on data which was not used to train the SVM. We note that the location of the same vesicle may vary slightly in $Q_\theta$ and $Q_{Exp}$, due to fluctuations in the manual labeling, for example. So we need to find the set $Q_{match}$, containing pairs $(r_1, r_2)$ with $r_1 \in Q_\theta$, $r_2 \in Q_{Exp}$, so that $r_1$ and $r_2$ are close to each other and describe the location of the same vesicle. We compute this with a simple, greedy but fast algorithm:

- compute the matrix $D_{ij} = \|r_i - r_j\|$ for all $r_i \in Q_\theta$, $r_j \in Q_{Exp}$
- while $D_{ij} = \min D \leq d_m$
    - put $(r_i, r_j)$ into $Q_{match}$
    - fill $i$-th row and $j$-th column of $D$ with $+\infty$

The resulting pairs of matching locations are closer than $d_m$, which should be set approximately to the radius of a vesicle. This algorithm does not generally find the global optimal assignment, which would be a NP-complete problem, but for low point densities the error made by this algorithm is usually low. Now we can evaluate the fraction of correctly detected and the fraction of false positives,

$$f_c = \frac{\#Q_{match}}{\#Q_{Exp}} \ , \qquad f_{fp} = 1 - \frac{\#Q_{match}}{\#Q_\theta} \ , \tag{12}$$

where $\#$ denotes the cardinality of the set. Depending on the threshold $\theta$, $\#Q_\theta$ may change and so does $\#Q_{match}$. So we get different values for $f_c$ and $f_{fp}$. We summarize these two rates in a diagram, which we call, following [10], *Receiver Operating Characteristic (ROC)*. In comparison to [10], $f_c$ represents the *hit rate* and $f_{fp}$ represents the *false alarm rate*, cf. Fig. 3. However, our ROC differs in some aspects. $f_c$ does not need to reach 1 for arbitrary low thresholds, as it is restricted by the set $Q_d$, which does not need to contain a match to all elements of $Q_{Exp}$. Furthermore, raising the threshold (decreasing $\#Q_\theta$) may occasionally increase $\#Q_{match}$ due to the greedy matching algorithm. These artifacts yield nonmonotonic parts in the ROC. If no a priori costs are assigned to $f_c$ and $f_{fp}$, then a natural measure for quality is the area below the ROC, which would be close to 1 at best, and 0 if no match would be contained in $Q_d$.

## 4.3 Results

The ROC of the validation with `ter04` and `mutant` is shown in Fig. 3. The rates $f_c$ and $f_{fp}$ were computed for 50 different threshold values, covering the interval $[\min_{r \in Q_d} z(r), \max_{r \in Q_d} z(r)]$. For `ter04` there exist four, and for `mutant` two, human expert labelings. Therefore we can plot either four or two curves, respectively, and get an impression about the variance of our performance measure, the area below the curve. Furthermore the multiple hand labelings allow us to plot them

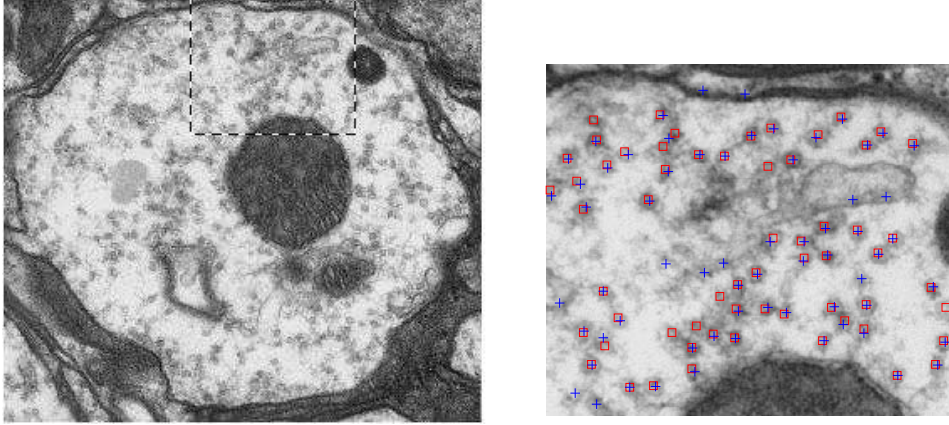

Figure 2: *left:* Photoreceptor terminal a of mutant type (`mutant`). *right:* Close up of the left panel, showing labels set by a human (+) and labels found by our method (□). Threshold $\theta$ was 0.3, which yields $f_c \approx 1 - f_{fp}$ in this case.

against each other in the same figure (single crosses). They indicate what performance is achievable at best. A curve passing these points can be considered to do the task as well on average as a human does. One can see that for the wild type image the curve gets close to that region. For the mutant the performance is slightly worse, in terms of the area. In mutants not only the number of vesicles, but typically also their shape and appearance differ. This variability was not covered by the training set and had to be generalized from the wild type data.

## 5    Discussion

We showed that SVR, used as a nonlinear filter, was able to detect synaptic vesicles in electron micrographs with high accuracy. On the one hand, for good performance the ability of the SVR to learn the input/output mapping properly is crucial. On the other hand it is necessary that in the input image a small neighborhood contains sufficient information about the target. Due to the "curse of dimensionality" (cf. [5]) the receptive field $P$ must not be too large, unless there is a huge amount of training data. A smaller input dimension $P$ would make the learning easier, but if $P$ is too small the information that $\mathbf{x}(r)$ contains about $y(r)$ may be too small and the performance poor. For the presented application patch size $P = 50 \times 50$ was a good tradeoff. Note that, since we do the filtering in the frequency domain, the size of $P$ has, in contrast to the number of SVs, no direct influence on the computation time needed for filtering. Thus, we have a 2500 dimensional input space and only 286 points in this space, that describe a vesicle. Clearly, only a model with low complexity would achieve acceptable generalization, and this is what we used. In fact the best linear SVR, i.e. the best linear filter, which has an even much lower complexity, still yields a performance of $A_{\mathtt{ter04}} = 0.82$ and $A_{\mathtt{mutant}} = 0.74$ (cf. Fig. 3, $A_{\mathtt{ter04}} = 0.85 \ldots 0.89$, $A_{\mathtt{ter04}} = 0.76 \ldots 0.83$). However, for future work we plan to extend the training set significantly. To do so, we have access to hand labelings for a broad variety of images of different mutants, also including slightly different scalings. With such more training data the nonlinear SVR can get more complex without loss of generalization performance. The capacity of the linear filter, however, cannot grow any further. Thus we expect the performance gap between nonlinear and linear filtering to grow significantly.

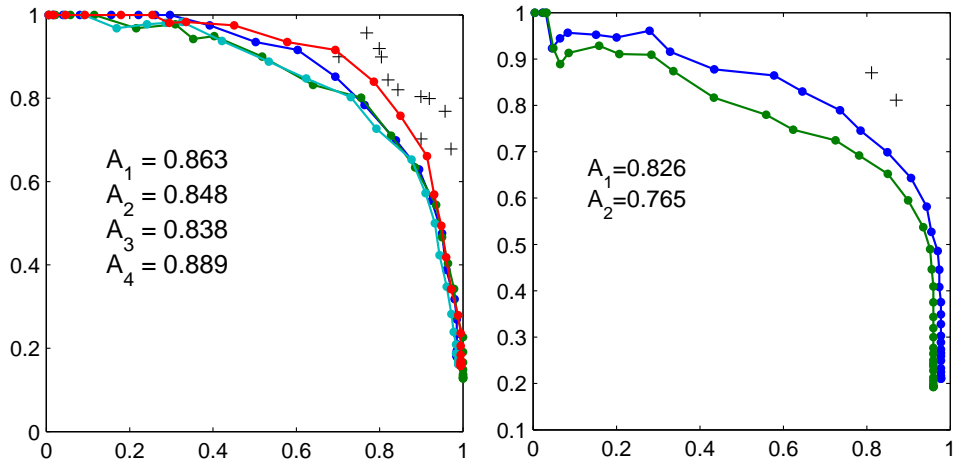

Figure 3: ROC of the validation with `ter04` *(left)* and with `mutant` *(right)*. For various thresholds $\theta$, $f_c$ is plotted on the x-axis versus $1 - f_{fp}$ on the y-axis. The single crosses show the fraction of matching labels for every pair of hand labels of `ter04`. For detailed explanation, see text.

## Acknowledgments

Support Contributed By: **BMBF** grant 0311559 (R.V., M.S., K.O.) and **NIH** grant EY-03592; Killam Trust (I.A.M.)

## References

[1] Bernhard Schölkopf and Alexander J. Smola. *Learning with Kernels*. The MIT Press, 2002.

[2] R.S. Stowers and T.L. Schwarz. A genetic method for generating drosophila eyes composed exclusively of mitotic clones of a single genotype. *Genetics*, (152):1631–1639, 1999.

[3] R. Fabian-Fine, P. Verstreken, P.R. Hiesinger, J.A. Horne, R. Kostyleva, H.J. Bellen, and I.A. Meinertzhagen. Endophilin acts after synaptic vesicle fission in drosophila photoreceptor terminals. *J. Neurosci.*, 2003. (in press).

[4] Simon S. Haykin. *Neural Networks: A Comprehensive Foundation*. Prentice Hall, 1998.

[5] Vladimir Vapnik. *The Nature of Statistical Learning Theory*. 1995.

[6] B. Schölkopf and A. Smola and R. Williamson and P. Bartlett. New support vector algorithms. *Neural Computation*, 12(5):1207–1245, May 2000.

[7] J. Mercer. Functions of positive and negative type and their connection with the theory of integral equations. *Philosophical Transactions of the Royal Society of London* **A**, 209:415–446, 1909.

[8] William H. Press, Saul A. Teukolsky, William T. Vetterling, and Brian P. Flannery. *Numerical Recipes in C*. Cambridge University Press, 2nd. edition, 1992.

[9] Chih-Chung Chang and Chih-Jen Lin. *LIBSVM – A Library for Support Vector Machines*. http://www.csie.ntu.edu.tw/~cjlin/libsvm/, April 2003.

[10] L. O. Harvey, Jr. The critical operating characteristic and the evaluation of expert judgment. *Organizational Behavior and Human Decision Processes*, 53(2):229–251, 1992.
